# Nearest Neighbor Based Feature Selection for Regression and its Application to Neural Activity

**Amir Navot**[12]   **Lavi Shpigelman**[12]   **Naftali Tishby**[12]   **Eilon Vaadia**[23]

[1]School of computer Science and Engineering
[2]Interdisciplinary Center for Neural Computation
[3]Dept. of Physiology, Hadassah Medical School
The Hebrew University Jerusalem, 91904, Israel

`Email for correspondence: {anavot,shpigi}@cs.huji.ac.il`

## Abstract

We present a non-linear, simple, yet effective, feature subset selection method for regression and use it in analyzing cortical neural activity. Our algorithm involves a *feature-weighted* version of the k-nearest-neighbor algorithm. It is able to capture complex dependency of the target function on its input and makes use of the leave-one-out error as a natural regularization. We explain the characteristics of our algorithm on synthetic problems and use it in the context of predicting hand velocity from spikes recorded in motor cortex of a behaving monkey. By applying feature selection we are able to improve prediction quality and suggest a novel way of exploring neural data.

## 1  Introduction

In many supervised learning tasks the input is represented by a very large number of features, many of which are not needed for predicting the labels. Feature selection is the task of choosing a small subset of features that is sufficient to predict the target labels well. Feature selection reduces the computational complexity of learning and prediction algorithms and saves on the cost of measuring non selected features. In many situations, feature selection can also enhance the prediction accuracy by improving the signal to noise ratio. Another benefit of feature selection is that the identity of the selected features can provide insights into the nature of the problem at hand. Therefore *feature selection* is an important step in efficient learning of large multi-featured data sets.

Feature selection (variously known as *subset selection*, *attribute selection* or *variable selection*) has been studied extensively both in statistics and by the machine learning community over the last few decades. In the most common selection paradigm an evaluation function is used to assign scores to subsets of features and a search algorithm is used to search for a subset with a high score. The evaluation function can be based on the performance of a specific predictor (*wrapper* model, [1]) or on some general (typically cheaper to compute) relevance measure of the features to the prediction (*filter* model). In any case, an exhaustive search over all feature sets is generally intractable due to the exponentially large number of possible sets. Therefore, search methods are employed which apply a variety of heuristics, such as hill climbing and genetic algorithms. Other methods simply rank individual features, assigning a score to each feature independently. These methods are usually very fast,

but inevitably fail in situations where only a combined set of features is predictive of the target function. See [2] for a comprehensive overview of feature selection and [3] which discusses selection methods for *linear* regression.

A possible choice of evaluation function is the leave-one-out (LOO) mean square error (MSE) of the *k-Nearest-Neighbor* (kNN) estimator ([4, 5]). This evaluation function has the advantage that it both gives a good approximation of the expected generalization error and can be computed quickly. [6] used this criterion on small synthetic problems (up to 12 features). They searched for good subsets using *forward selection*, *backward elimination* and an algorithm (called *schemata)* that *races* feature sets against each other (eliminating poor sets, keeping the fittest) in order to find a subset with a good score. All these algorithms perform a local search by flipping one or more features at a time. Since the space is discrete the direction of improvement is found by trial and error, which slows the search and makes it impractical for large scale real world problems involving many features.

In this paper we develop a novel selection algorithm. We extend the LOO-kNN-MSE evaluation function to assign scores to *weight vectors* over the features, instead of just to feature subsets. This results in a smooth ("almost everywhere") function over a continuous domain, which allows us to compute the gradient analytically and to employ a stochastic gradient ascent to find a locally optimal weight vector. The resulting weights provide a ranking of the features, which we can then threshold in order to produce a subset. In this way we can apply an easy-to-compute, gradient directed search, without relearning of a regression model at each step but while employing a strong non-linear function estimate (kNN) that can capture complex dependency of the function on its features[1].

Our motivation for developing this method is to address a major computational neuroscience question: which features of the neural code are relevant to the observed behavior. This is an important element of enabling interpretability of neural activity. Feature selection is a promising tool for this task. Here, we apply our feature selection method to the task of reconstructing hand movements from neural activity, which is one of the main challenges in implementing brain computer interfaces [8]. We look at neural population spike counts, recorded in motor cortex of a monkey while it performed hand movements and locate the most informative subset of neural features. We show that it is possible to improve prediction results by wisely selecting a subset of cortical units and their time lags, relative to the movement. Our algorithm, which considers feature subsets, outperforms methods that consider features on an individual basis, suggesting that complex dependency on a set of features exists in the code.

The remainder of the paper is organized as follows: we describe the problem setting in section 2. Our method is presented in section 3. Next, we demonstrate its ability to cope with a complicated dependency of the target function on groups of features using synthetic data (section 4). The results of applying our method to the hand movement reconstruction problem is presented in section 5.

## 2  Problem Setting

First, let us introduce some notation. Vectors in $R^n$ are denoted by boldface small letters (e.g. $\mathbf{x}$, $\mathbf{w}$). Scalars are denoted by small letters (e.g. $x$, $y$). The $i$'th element of a vector $\mathbf{x}$ is denoted by $x_i$. Let $f(\mathbf{x})$, $f : R^n \longrightarrow R$ be a function that we wish to estimate. Given a set $S \subset R^n$, the empiric *mean square error* (MSE) of an estimator $\hat{f}$ for $f$ is defined as $MSE_S(\hat{f}) = \frac{1}{|S|} \sum_{\mathbf{x} \in S} \left( f(\mathbf{x}) - \hat{f}(\mathbf{x}) \right)^2$.

**kNN Regression** *k-Nearest-Neighbor* (kNN) is a simple, intuitive and efficient way to estimate the value of an unknown function in a given point using its values in other (training) points. Let $S = \{\mathbf{x}_1, \ldots, \mathbf{x}_m\}$ be a set of training points. The kNN estimator is defined as the mean function value of the nearest neighbors: $\hat{f}(\mathbf{x}) = \frac{1}{k} \sum_{\mathbf{x}' \in N(\mathbf{x})} f(x')$ where $N(\mathbf{x}) \subset S$ is the set of $k$ nearest points to $\mathbf{x}$ in $S$ and $k$ is a parameter([4, 5]). A softer version takes a *weighted* average, where the weight of each neighbor is proportional to its proximity. One specific way of doing this is

$$\hat{f}(\mathbf{x}) = \frac{1}{Z} \sum_{\mathbf{x}' \in N(\mathbf{x})} f(\mathbf{x}') e^{-d(\mathbf{x}, \mathbf{x}')/\beta} \tag{1}$$

where $d(\mathbf{x}, \mathbf{x}') = \|\mathbf{x} - \mathbf{x}'\|_2^2$ is the $\ell_2$ norm, $Z = \sum_{\mathbf{x}' \in N(\mathbf{x})} e^{-d(\mathbf{x}, \mathbf{x}')/\beta}$ is a normalization factor and $\beta$ is a parameter. The soft kNN version will be used in the remainder of this paper. This regression method is a special form of *locally weighted regression* (See [5] for an overview of the literature on this subject.) It has the desirable property that no learning (other than storage of the training set) is required for the regression. Also note that the Gaussian Radial Basis Function has the form of a *kernel* ([9]) and can be replaced with any operator on two data points that decays as a function of the difference between them (e.g. kernel induced distances). As will be seen in the next section, we use the MSE of a modified kNN regressor to guide the search for a set of features $F \subset \{1, \ldots n\}$ that achieves a low MSE. However, the MSE and the Gaussian kernel can be replaced by other loss measures and kernels (respectively) as long as they are differentiable almost everywhere.

## 3   The Feature Selection Algorithm

In this section we present our selection algorithm called *RGS* (Regression, Gradient guided, feature Selection). It can be seen as a filter method for general regression algorithms or as a wrapper for estimation by the kNN algorithm.

Our goal is to find subsets of features that induce a small estimation error. As in most supervised learning problems, we wish to find subsets that induce a small generalization error, but since it is not known, we use an *evaluation function* on the training set. This evaluation function is defined not only for subsets but for any weight vector over the features. This is more general because a feature subset can be represented by a binary weight vector that assigns a value of one to features in the set and zero to the rest of the features.

For a given weights vector over the features $\mathbf{w} \in R^n$, we consider the weighted squared $\ell_2$ norm induced by $\mathbf{w}$, defined as $\|z\|_{\mathbf{w}}^2 = \sum_i z_i^2 w_i^2$. Given a training set $S$, we denote by $\hat{f}_{\mathbf{w}}(\mathbf{x})$ the value assigned to $\mathbf{x}$ by a weighted kNN estimator, defined in equation 1, using the weighted squared $\ell_2$-norm as the distances $d(\mathbf{x}, \mathbf{x}')$ and the nearest neighbors are found among the points of $S$ excluding $\mathbf{x}$. The evaluation function is defined as the negative (halved) square error of the weighted kNN estimator:

$$e(\mathbf{w}) = -\frac{1}{2} \sum_{\mathbf{x} \in S} \left( f(\mathbf{x}) - \hat{f}_w(\mathbf{x}) \right)^2. \tag{2}$$

This evaluation function scores weight vectors ($\mathbf{w}$). A change of weights will cause a change in the distances and, possibly, the identity of each point's nearest neighbors, which will change the function estimates. A weight vector that induces a distance measure in which neighbors have similar labels would receive a high score. The mean, $1/|S|$ is replaced with a $1/2$ to ease later differentiation. Note that there is no explicit regularization term in $e(\mathbf{w})$. This is justified by the fact that for each point, the estimate of its function value does not include that point as part of the training set. Thus, equation 2 is a leave-one-out cross validation error. Clearly, it is impossible to go over all the weight vectors (or even over all the feature subsets), and therefore some search technique is required.

**Algorithm 1** $RGS(S, k, \beta, T)$

1. initialize $\mathbf{w} = (1, 1, \ldots, 1)$
2. for $t = 1 \ldots T$

   (a) pick randomly an instance $\mathbf{x}$ from $S$

   (b) calculate the gradient of $e(\mathbf{w})$:

   $$
   \begin{aligned}
   \nabla e(\mathbf{w}) &= -\sum_{\mathbf{x} \in S} \left( f(\mathbf{x}) - \hat{f}_w(\mathbf{x}) \right) \nabla_{\mathbf{w}} \hat{f}_{\mathbf{w}}(\mathbf{x}) \\
   \nabla_{\mathbf{w}} \hat{f}_{\mathbf{w}}(\mathbf{x}) &= \frac{-\frac{4}{\beta} \sum_{\mathbf{x}'', \mathbf{x}' \in N(\mathbf{x})} f(x'') a(x', x'') \, \mathbf{u}(x', x'')}{\sum_{\mathbf{x}'', \mathbf{x}' \in N(\mathbf{x})} a(x', x'')}
   \end{aligned}
   $$

   where $a(x', x'') = e^{-\left( ||x - x'||_{\mathbf{w}}^2 + ||x - x''||_{\mathbf{w}}^2 \right)/\beta}$
   and $\mathbf{u}(x', x'') \in R^n$ is a vector with $u_i = w_i \left[ (x_i - x_i')^2 + (x_i - x_i'')^2 \right]$.

   (c) $\mathbf{w} = \mathbf{w} + \eta_t \nabla e(\mathbf{w}) = \mathbf{w} \left( 1 + \eta_t \nabla_{\mathbf{w}} \hat{f}_{\mathbf{w}}(\mathbf{x}) \right)$ where $\eta_t$ is a decay factor.

Our method finds a weight vector $\mathbf{w}$ that locally maximizes $e(\mathbf{w})$ as defined in (2) and then uses a threshold in order to obtain a feature subset. The threshold can be set either by cross validation or by finding a natural cutoff in the weight values. However, we later show that using the distance measure induced by $\mathbf{w}$ in the regression stage compensates for taking too many features. Since $e(\mathbf{w})$ is defined over a continuous domain and is smooth almost everywhere we can use gradient ascent in order to maximize it. *RGS* (algorithm 1) is a stochastic gradient ascent over $e(\mathbf{w})$. In each step the gradient is evaluated using one sample point and is added to the current weight vector. *RGS* considers the weights of all the features at the same time and thus it can handle dependency on a group of features. This is demonstrated in section 4. In this respect, it is superior to selection algorithms that scores each feature independently. It is also faster than methods that try to find a good subset directly by trial and error. Note, however, that convergence to global optima is not guaranteed and standard techniques to avoid local optima can be used.

The parameters of the algorithm are $k$ (number of neighbors), $\beta$ (Gaussian decay factor), $T$ (number of iterations) and $\{\eta_t\}_{t=1}^{T}$ (step size decay scheme). The value of $k$ can be tuned by cross validation, however a proper choice of $\beta$ can compensate for a $k$ that is too large. It makes sense to tune $\beta$ to a value that places most neighbors in an active zone of the Gaussian. In our experiments, we set $\beta$ to half of the mean distance between points and their $k$ neighbors. It usually makes sense to use $\eta_t$ that decays over time to ensure convergence, however, on our data, convergence was also achieved with $\eta_t = 1$.

The computational complexity of *RGS* is $\Theta(TNm)$ where $T$ is the number of iterations, $N$ is the number of features and $m$ is the size of the training set $S$. This is correct for a naive implementation which finds the nearest neighbors and their distances from scratch at each step by measuring the distances between the current point to all the other points. RGS is basically an on line method which can be used in batch mode by running it in epochs on the training set. When it is run for only one epoch, $T = m$ and the complexity is $\Theta\left(m^2 N\right)$. Matlab code for this algorithm (and those that we compare with) is available at
http://www.cs.huji.ac.il/labs/learning/code/fsr/

## 4 Testing on synthetic data

The use of synthetic data, where we can control the importance of each feature, allows us to illustrate the properties of our algorithm. We compare our algorithm with other common

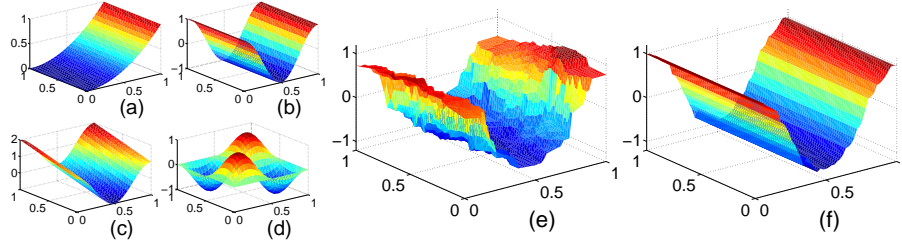

Figure 1: (a)-(d): Illustration of the four synthetic target functions. The plots shows the functions value as function of the first two features. (e),(f): demonstration of the effect of feature selection on estimating the second function using kNN regression ($k = 5$, $\beta = 0.05$). (e) using both features ($mse = 0.03$), (f) using the relevant feature only ($mse = 0.004$)

selection methods: *infoGain* [10], correlation coefficients (*corrcoef*) and *forward selection* (see [2]) . *infoGain* and *corrcoef* simply rank features according to the mutual information[2] or the correlation coefficient (respectively) between each feature and the labels (i.e. the target function value). Forward selection (*fwdSel*) is a greedy method in which features are iteratively added into a growing subset. In each step, the feature showing the greatest improvement (given the previously selected subset) is added. This is a search method that can be applied to any evaluation function and we use our criterion (equation 2 on feature subsets). This well known method has the advantages of considering feature subsets and that it can be used with non linear predictors. Another algorithm we compare with scores each feature independently using our evaluation function (2). This helps us in analyzing *RGS*, as it may help single out the respective contributions to performance of the properties of the evaluation function and the search method. We refer to this algorithm as *SKS* (Single feature, kNN regression, feature Selection).

We look at four different target functions over $R^{50}$. The training sets include 20 to 100 points that were chosen randomly from the $[-1, 1]^{50}$ cube. The target functions are given in the top row of figure 2 and are illustrated in figure 1(a-d). A random Gaussian noise with zero mean and a variance of $1/7$ was added to the function value of the training points. Clearly, only the first feature is relevant for the first two target functions, and only the first two features are relevant for the last two target functions. Note also that the last function is a smoothed version of parity function learning and is considered hard for many feature selection algorithms [2].

First, to illustrate the importance of feature selection on regression quality we use kNN to estimate the second target function. Figure 1(e-f) shows the regression results for target (b), using either only the relevant feature or both the relevant and an irrelevant feature. The addition of one irrelevant feature degrades the MSE ten fold. Next, to demonstrate the capabilities of the various algorithms, we run them on each of the above problems with varying training set size. We measure their success by counting the number of times that the relevant features were assigned the highest rank (repeating the experiment 250 times by re-sampling the training set). Figure 2 presents success rate as function of training set size. We can see that all the algorithms succeeded on the first function which is monotonic and depends on one feature alone. *infoGain* and *corrcoef* fail on the second, non-monotonic function. The three kNN based algorithms succeed because they only depend on local properties of the target function. We see, however, that RGS needs a larger training set to achieve a high success rate. The third target function depends on two features but the dependency is simple as each of them alone is highly correlated with the function value. The fourth, XOR-like function exhibits a complicated dependency that requires consideration of the two relevant features simultaneously. *SKS* which considers features separately sees the effect of all other features as noise and, therefore, has only marginal success on the third

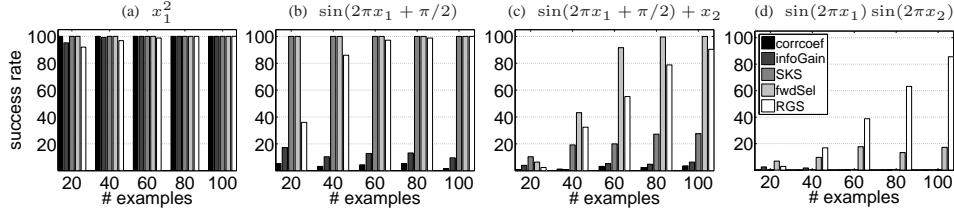

Figure 2: Success rate of the different algorithms on 4 synthetic regression tasks (averaged over 250 repetitions) as a function of the number of training examples. Success is measured by the percent of the repetitions in which the relevant feature(s) received first place(s).

function and fails on the fourth altogether. *RGS* and *fwdSel* apply different search methods. *fwdSel* considers subsets but can evaluate only one additional feature in each step, giving it some advantage over *RGS* on the third function but causing it to fail on the fourth. *RGS* takes a step in all features simultaneously. Only such an approach can succeed on the fourth function.

## 5 Hand Movements Reconstruction from Neural Activity

To suggest an interpretation of neural coding we apply *RGS* and compare it with the alternatives presented in the previous section[3] on the hand movement reconstruction task. The data sets were collected while a monkey performed a planar center-out reaching task with one or both hands [11]. 16 electrodes, inserted daily into novel positions in primary motor cortex were used to detect and sort spikes in up to 64 channels (4 per electrode). Most of the channels detected isolated neuronal spikes by template matching. Some, however, had templates that were not tuned, producing spikes during only a fraction of the session. Others (about 25%) contained unused templates (resulting in a constant zero producing channel or, possibly, a few random spikes). The rest of the channels (one per electrode) produced spikes by threshold passing. We construct a labeled regression data set as follows. Each example corresponds to one time point in a trial. It consists of the spike counts that occurred in the 10 previous consecutive $100ms$ long time bins from all 64 channels ($64 \times 10 = 640$ features) and the label is the X or Y component of the instantaneous hand velocity. We analyze data collected over 8 days. Each data set has an average of 5050 examples collected during the movement periods of the successful trials.

In order to evaluate the different feature selection methods we separate the data into training and test sets. Each selection method is used to produce a ranking of the features. We then apply kNN (based on the training set) using different size groups of top ranking features to the test set. We use the resulting MSE (or correlation coefficient between true and estimated movement) as our measure of quality. To test the significance of the results we apply 5-fold cross validation and repeat the process 5 times on different permutations of the trial ordering. Figure 3 shows the average (over permutations, folds and velocity components) MSE as a function of the number of selected features on four of the different data sets (results on the rest are similar and omitted due to lack of space)[4]. It is clear that *RGS* achieves better results than the other methods throughout the range of feature numbers.

To test whether the performance of *RGS* was consistently better than the other methods we counted winning percentages (the percent of the times in which *RGS* achieved lower MSE than another algorithm) in all folds of all data sets and as a function of the number of

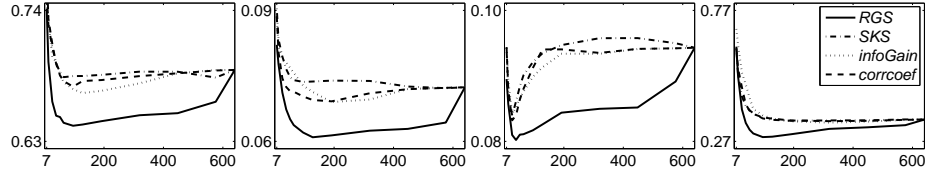

Figure 3: MSE results for the different feature selection methods on the neural activity data sets. Each sub figure is a different recording day. MSEs are presented as a function of the number of features used. Each point is a mean over all 5 cross validation folds, 5 permutations on the data and the two velocity component targets. Note that some of the data sets are harder than others.

features used. Figure 4 shows the winning percentages of *RGS* versus the other methods. For a very low number of features, while the error is still high, *RGS* winning scores are only slightly better than chance but once there are enough features for good predictions the winning percentages are higher than 90%. In figure 3 we see that the MSE achieved when using only approximately 100 features selected by *RGS* is better than when using all the features. This difference is indeed statistically significant (win score of 92%). If the MSE is replaced by correlation coefficient as the measure of quality, the average results (not shown due to lack of space) are qualitatively unchanged.

*RGS* not only ranks the features but also gives them weights that achieve locally optimal results when using kNN regression. It therefore makes sense not only to select the features but to weigh them accordingly. Figure 5 shows the winning percentages of *RGS* using the weighted features versus *RGS* using uniformly weighted features. The corresponding MSEs (with and without weights) on the first data set are also displayed. It is clear that using the weights improves the results in a manner that becomes increasingly significant as the number of features grows, especially when the number of features is greater than the optimal number. Thus, using weighted features can compensate for choosing too many by diminishing the effect of the surplus features.

To take a closer look at what features are selected, figure 6 shows the 100 highest ranking features for all algorithms on one data set. Similar selection results were obtained in the rest of the folds. One would expect to find that well isolated cells (template matching) are more informative than threshold based spikes. Indeed, all the algorithms select isolated cells more frequently within the top 100 features (*RGS* does so in 95% of the time and the rest in 70%-80%). A human selection of channels, based only on looking at raster plots and selecting channels with stable firing rates was also available to us. This selection was independent of the template/threshold categorisation. Once again, the algorithms selected the humanly preferred channels more frequently than the other channels. Another and more interesting observation that can also be seen in the figure is that while *corrcoef, SKS* and *infoGain* tend to select all time lags of a channel, *RGS*'s selections are more scattered (more channels and only a few time bins per channel). Since *RGS* achieves best results, we

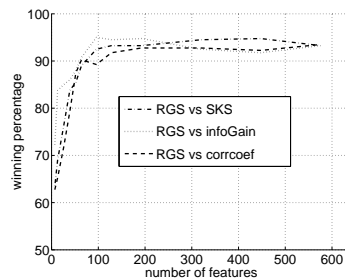

Figure 4: Winning percentages of *RGS* over the other algorithms. *RGS* achieves better MSEs consistently.

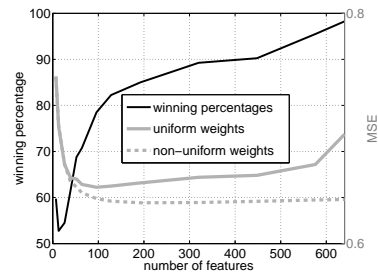

Figure 5: Winning percentages of *RGS* with and without weighting of features (black). Gray lines are corresponding MSEs of these methods on the first data set.

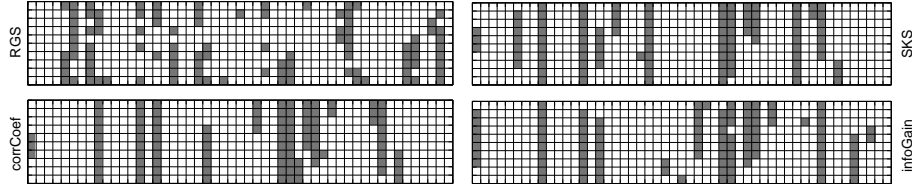

Figure 6: 100 highest ranking features (grayed out) selected by the algorithms. Results are for one fold of one data set. In each sub figure the bottom row is the ($100ms$) time bin with least delay and the higher rows correspond to longer delays. Each column is a channel (silent channels omitted).

conclude that this selection pattern is useful. Apparently *RGS* found these patterns thanks to its ability to evaluate complex dependency on feature subsets. This suggests that such dependency of the behavior on the neural activity does exist.

## 6 Summary

In this paper we present a new method of selecting features for function estimation and use it to analyze neural activity during a motor control task . We use the leave-one-out mean squared error of the kNN estimator and minimize it using a gradient ascent on an "almost" smooth function. This yields a selection method which can handle a complicated dependency of the target function on groups of features yet can be applied to large scale problems. This is valuable since many common selection methods lack one of these properties. By comparing the result of our method to other selection methods on the motor control task, we show that consideration of complex dependency helps to achieve better performance. These results suggest that this is an important property of the code.

Our future work is aimed at a better understanding of neural activity through the use of feature selection. One possibility is to perform feature selection on other kinds of neural data such as local field potentials or retinal activity. Another promising option is to explore the temporally changing properties of neural activity. Motor control is a dynamic process in which the input output relation has a temporally varying structure. *RGS* can be used in on line (rather than batch) mode to identify these structures in the code.

## Footnotes

[1]The design of this algorithm was inspired by work done by Gilad-Bachrach et al. ([7]) which used a large margin based evaluation function to derive feature selection algorithms for classification.

[2]Feature and function values were "binarized" by comparing them to the median value.

[3]*fwdSel* was not applied due to its intractably high run time complexity. Note that its run time is at least $r$ times that of *RGS* where $r$ is the size of the optimal set and is longer in practice.

[4]We use $k = 50$ (approximately 1% of the data points). $\beta$ is set automatically as described in section 3. These parameters were manually tuned for good kNN results and were not optimized for any of the feature selection algorithms. The number of epochs for *RGS* was set to 1 (i.e. $T = m$).

## References

[1] R. Kohavi and G.H. John. Wrapper for feature subset selection. *Artificial Intelligence*, 97(1-2):273–324, 1997.

[2] I. Guyon and A. Elisseeff. An introduction to variable and feature selection. *JMLR*, 2003.

[3] A.J. Miller. *Subset Selection in Regression*. Chapman and Hall, 1990.

[4] L. Devroye. The uniform convergence of nearest neighbor regression function estimators and their application in optimization. *IEEE transactions in information theory*, 24(2), 1978.

[5] C. Atkeson, A. Moore, and S. Schaal. Locally weighted learning. *AI Review*, 11.

[6] O. Maron and A. Moore. The racing algorithm: Model selection for lazy learners. In *Artificial Intelligence Review*, volume 11, pages 193–225, April 1997.

[7] R. Gilad-Bachrach, A. Navot, and N. Tishby. Margin based feature selection - theory and algorithms. In *Proc. $21^{st}$ (ICML)*, pages 337–344, 2004.

[8] D. M. Taylor, S. I. Tillery, and A. B. Schwartz. Direct cortical control of 3d neuroprosthetic devices. *Science*, 296(7):1829–1832, 2002.

[9] V. Vapnik. *The Nature Of Statistical Learning Theory*. Springer-Verlag, 1995.

[10] J. R. Quinlan. Induction of decision trees. In Jude W. Shavlik and Thomas G. Dietterich, editors, *Readings in Machine Learning*. Morgan Kaufmann, 1990. Originally published in *Machine Learning* 1:81–106, 1986.

[11] R. Paz, T. Boraud, C. Natan, H. Bergman, and E. Vaadia. Preparatory activity in motor cortex reflects learning of local visuomotor skills. *Nature Neuroscience*, 6(8):882–890, August 2003.
